# A Randomized Algorithm for Pairwise Clustering

**Yoram Gdalyahu, Daphna Weinshall, Michael Werman**
Institute of Computer Science, The Hebrew University, 91904 Jerusalem, Israel
{yoram,daphna,werman}@cs.huji.ac.il

## Abstract

We present a stochastic clustering algorithm based on pairwise similarity of datapoints. Our method extends existing deterministic methods, including agglomerative algorithms, min-cut graph algorithms, and connected components. Thus it provides a common framework for all these methods. Our graph-based method differs from existing stochastic methods which are based on analogy to physical systems. The stochastic nature of our method makes it more robust against noise, including accidental edges and small spurious clusters. We demonstrate the superiority of our algorithm using an example with 3 spiraling bands and a lot of noise.

## 1 Introduction

Clustering algorithms can be divided into two categories: those that require a vectorial representation of the data, and those which use only pairwise representation. In the former case, every data item must be represented as a vector in a real normed space, while in the second case only pairwise relations of similarity or dissimilarity are used. The pairwise information can be represented by a weighted graph $G(V, E)$: the nodes $V$ represent data items, and the positive weight $w_{ij}$ of an edge $(i, j)$ representing the amount of similarity or dissimilarity between items $i$ and $j$. The graph $G$ might not be a complete graph. In the rest of this paper $w_{ij}$ represents a similarity value.

A vectorial representation is very convenient when one has either an explicit or an implicit parametric model for the data. An implicit model means that the data distribution function is not known, but it is assumed, e.g., that every cluster is symmetrically distributed around some center. An explicit model specifically describes the *shape* of the distribution (e.g., Gaussian). In these cases, if a vectorial representation is available, the clustering procedure may rely on iterative estimation of means (e.g., [2, 8]).

In the absence of a vectorial representation, one can either try to embed the graph of distances in a vector space, or use a direct pairwise clustering method. The

embedding problem is difficult, since it is desirable to use a representation that is both low dimensional and has a low distortion of distances [6, 7, 3]. Moreover, even if such embedding is achieved, it can help to cluster the data only if at least an implicit parametric model is valid. Hence, direct methods for pairwise clustering are of great value.

One strategy of pairwise clustering is to use a similarity threshold $\theta$, remove edges with weight less than $\theta$, and identify the connected components that remain as clusters. A transformation of weights may precede the thresholding[1]. The physically motivated transformation in [1] uses a granular magnet model and replaces weights by "spin correlations". Our algorithm is similar to this model, see Section 2.4.

A second pairwise clustering strategy is used by agglomerative algorithms [2], which start with the trivial partition of $N$ points into $N$ clusters of size one, and continue by subsequently merging pairs of clusters. At every step the two clusters which are most similar are merged together, until the similarity of the closest clusters is lower than some threshold. Different similarity measures between clusters distinguish between different agglomerative algorithms. In particular, the single linkage algorithm defines the similarity between clusters as the maximal similarity between two of their members, and the complete linkage algorithm uses the minimal value.

A third strategy of pairwise clustering uses the notion of cuts in a graph. A cut $(A, B)$ in a graph $G(V, E)$ is a partition of $V$ into two disjoint sets $A$ and $B$. The capacity of the cut is the sum of weights of all edges that cross the cut, namely: $c(A, B) = \sum_{i \in A, j \in B} w_{ij}$. Among all the cuts that separate two marked vertices, the *minimal cut* is the one which has minimal capacity. The minimal cut clustering algorithm [11] divides the graph into components using a cascade of minimal cuts[2].

The normalized cut algorithm [9] uses the association of $A$ (sum of weights incident on $A$) and the association of $B$ to normalize the capacity $c(A, B)$. In contrast with the easy min-cut problem, the problem of finding a minimal normalized cut (Ncut) is NP-hard, but with certain approximations it reduces to a generalized eigenvalue problem [9].

Other pairwise clustering methods include techniques of non parametric density estimation [4] and pairwise deterministic annealing [3]. However, the three categories of methods above are of special importance to us, since our current work provides a common framework for all of them. Specifically, our new algorithm may be viewed as a randomized version of an agglomerative clustering procedure, and in the same time it generalizes the minimal cut algorithm. It is also strongly related to the physically motivated granular magnet model algorithm. By showing the connection between these methods, which may seem very different at a first glance, we provide a better understanding of pairwise clustering.

Our method is unique in its stochastic nature while provenly maintaining low complexity. Thus our method performs as well as the aforementioned methods in "easy" cases, while keeping the good performance in "difficult" cases. In particular, it is more robust against noise and pathological configurations: (i) A *minimal cut* algorithm is intuitively reasonable since it optimizes so that as much of the similarity

weight remains within the parts of the clusters, and as little as possible is "wasted" between the clusters. However, it tends to fail when there is no clean separation into 2 parts, or when there are many small spurious parts due, e.g., to noise. Our stochastic approach avoids these problems and behaves more robustly. (ii) The *single linkage* algorithm deals well with chained data, where items in a cluster are connected by transitive relations. Unfortunately the deterministic construction of chains can be harmful in the presence of noise, where a few points can make a "bridge" between two large clusters and merge them together. Our algorithm inherits the ability to cluster chained data; at the same time it is robust against such noisy bridges as long as the probability to select all the edges in the bridge remains small.

## 2 Stochastic pairwise clustering

Our randomized clustering algorithm is constructed of two main steps:

1. Stochastic partition of the similarity graph into $r$ parts (by randomized agglomeration). For each partition index $r$ ($r = N \ldots 1$):

   (a) for every pair of points, the probability that they remain in the same part is computed;

   (b) the weight of the edge between the two points is replaced by this probability;

   (c) clusters are formed using connected components and threshold of 0.5.

   This is described in Sections 2.1 and 2.2.

2. Selection of proper $r$ values, which reflect "interesting" structure in our problem. This is described in Section 2.3.

### 2.1 The similarity transformation

At each level $r$, our algorithm performs a similarity transformation followed by thresholding. In introducing this process, our starting point is a generalization of the minimal cut algorithm; then we show how this generalization is obtained by the randomization of a single linkage algorithm.

First, instead of considering only the minimal cuts, let us induce a probability distribution on the set of *all* cuts. We assign to each cut a probability which decreases with increasing capacity. Hence the minimal cut is the most probable cut in the graph, but it does not determine the graph partition on its own.

As a second generalization to the min-cut algorithm we consider multi-way cuts. An *r-way cut* is a partition of $G$ into $r$ connected components. The capacity of an $r$-way cut is the sum of weights of all edges that connect different components. In the rest of this paper we may refer to $r$-way cuts simply as "cuts".

Using the distribution induced on $r$-way cuts, we apply the following family of *weight transformations*. The weight $w_{ij}$ is replaced by the probability that nodes $i$ and $j$ are in the same side of a random $r$-way cut: $w_{ij} \to p_{ij}^r$. This transformation is defined for every integer $r$ between 1 and $N$.

Since the number of cuts in a graph is exponentially large, one must ask whether $p_{ij}^r$ is computable. Here the decaying rate of the cut probability plays an essential role. The induced probability is found to decay fast enough with the capacity, hence $p_{ij}^r$ is dominated by the low capacity cuts. Thus, since there exists a polynomial

427 (r-1)

bound on the number of low capacity cuts in any graph [5], the problem becomes computable.

This strong property suggests a sampling scheme to estimate the pairing probabilities. Assume that a sampling tool is available, which generates cuts according to their probability. Under this condition, a sample of polynomial size is sufficient to estimate the $p_{ij}^r$'s.

The sampling tool that we use is called the "contraction algorithm" [5]. Its discovery led to an efficient probabilistic algorithm for the minimal cut problem. It was shown that for a given $r$, the probability that the contraction algorithm returns the minimal $r$-way cut of any graph is at least $N^{-2(r-1)}$, and it decays with increasing capacity[3]. For a graph which is really made of clusters this is a rough underestimation.

The contraction algorithm can be implemented in several ways. We describe here its simplest form, which is constructed from $N$-1 *edge contraction* steps. Each edge contraction follows the procedure below:

- Select edge $(i, j)$ with probability proportional to $w_{ij}$.
- Replace nodes $i$ and $j$ by a single node $\{ij\}$.
- Let the set of edges incident on $\{ij\}$ be the union of the sets of edges incident on $i$ and $j$, but remove self loops formed by edges originally connecting $i$ to $j$.

It is shown in [5] that each step of edge contraction can be implemented in $O(N)$ time, hence this simple form of the contraction algorithm has complexity of $O(N^2)$. For sparse graphs an $O(N \log N)$ implementation can be shown.

The contraction algorithm as described above is a randomized version of the agglomerative single linkage procedure. If the probabilistic selection rule is replaced by a greedy selection of the maximal weight edge, the single linkage algorithm is obtained.

In terms of similarity transformations, a single linkage algorithm which halts with $r$ clusters may be associated with the transformation $w_{ij} \rightarrow 0, 1$ (1 if $i$ and $j$ are returned at the same cluster, 0 otherwise). Our similarity transformation $(p_{ij}^r)$ uses the expected value (or the average) of of this binary assignment under the probabilistic relaxation of the selection rule.

We could estimate $p_{ij}^r$ by repeating the contraction algorithm $M$ times and averaging these binary indicators (a better way is described below). Using Chernoff inequality it can be shown[4] that if $M \geq (2 \ln 2 + 4 \ln N - 2 \ln \delta)/\epsilon^2$ then each $p_{ij}^r$ is estimated, with probability $\geq 1 - \delta$, within $\epsilon$ from its true value.

## 2.2 Construction of partitions

To compute a partition at every $r$ level, it is sufficient to know for every $i$-$j$ pair which $r$ satisfies $p_{ij}^r = 0.5$.

This is found by repeating the contraction algorithm $M$ times. In each iteration there exists a single $r$ at which the edge between points $i - j$ is marked and the points are merged. Denote by $r_m$ the level $r$ which joins $i$ and $j$ at the $m$-$th$ iteration $(m = 1 \ldots M)$. The median $r'$ of the sequence $\{r_1, r_2 \ldots r_M\}$ is the sample estimate

for the level $r$ that satisfies $p_{ij}^r = 0.5$. We use an on-line technique (not described here) to estimate the median $r'$ using constant and small memory.

Having computed the matrix $r'$, where the entry $r'_{ij}$ is the estimator for $r$ that satisfies $p_{ij}^r = 0.5$, we find the connected components at a given $r$ value after disconnecting every edge $(i,j)$ for which $r'_{ij} > r$. This gives the $r$ level partition.

## 2.3   Hierarchical clustering

We now address the problem of choosing "good" $r$ values.

The transformed weight $p_{ij}^r$ has the advantage of reflecting transitive relations between data items $i$ and $j$. For a selected value of $r$ (which defines a specification level) the partition of data items into clusters is obtained by eliminating edges whose weight $(p_{ij}^r)$ is less than a fixed threshold (0.5). That is: nodes are assigned to the same cluster if at level $r$ their probability to be on the same side of a random r-way cut is larger than half.

Partitions which correspond to subsequent $r$ values might be very similar to each other, or even identical, in the sense that only a few nodes (if at all) change the component to which they belong. Events which are of interest, therefore, are when the variation between subsequent partitions is of the order of the size of a cluster. This typically happens when two clusters combine to form one cluster which corresponds to a higher scale (less resolution).

In accordance, using the hierarchical partition obtained in Section 2.2, we measure the variation between subsequent partitions by $\sum_{k=1}^{K} \Delta N_k$, where $K$ is a small constant (of the order of the number of clusters) and $N_k$ is the size of the $k^{th}$ largest component of the partition.

## 2.4   The granular magnet model

Our algorithm is closely related to the successful granular magnet model recently proposed in [1]. However, the two methods draw the random cuts effectively from different distributions. In our case the distribution is data driven, imposed by the contraction algorithm. The physical model imposes the Boltzmann distribution, where a cut of capacity $E$ is assigned a probability proportional to $\exp(-E/T)$, and $T$ is a temperature parameter.

The probability $p_{ij}^T$ measures whether nodes $i$ and $j$ are on the same side of a cut at temperature $T$ (originally called "spin-spin correlation function"). The magnetic model uses the similarity transformation $w_{ij} \to p_{ij}^T$ and a threshold (0.5) to break the graph into components. However, even if identical distributions were used, $p_{ij}^T$ is inherently different from $p_{ij}^r$ since at a fixed temperature the random cuts may have different numbers of components.

Superficially, the parameter $T$ plays in the magnetic model a similar role to our parameter $r$. But the two parameterizations are quite different. First, $r$ is a discrete parameter while $T$ is a continuous one. Moreover, in order to find the pairing probabilities $p_{ij}^T$ for different temperatures, the stochastic process should be employed for every $T$ value separately. On the other hand, our algorithm estimates $p_{ij}^r$ for every $1 \le r \le N$ at once. For hard clustering (v.s. soft clustering) it was shown above that even this is not necessary, since we can get a direct estimation of $r$ which satisfies $p_{ij}^r = 0.5$.

# 3  Example

Pairwise clustering has the advantage that a vectorial representation of the data is not needed. However, graphs of distances are hard to visualize and we therefore demonstrate our algorithm using vectorial data. In spite of having vectorial representation, the information which is made available to the clustering algorithm includes only the matrix of pairwise Euclidean distances[5] $d_{ij}$. Since our algorithm works with similarity values and not with distances, it is necessary to invert the distances using $w_{ij} = f(d_{ij})$. We choose $f$ to be similar to the function used in [1]: $w_{ij} = \exp(-d_{ij}^2/a^2)$ where $a$ is the average distance to the $n$-th nearest neighbor (we used $n=10$, but the results remain the same as long as a reasonable value is selected).

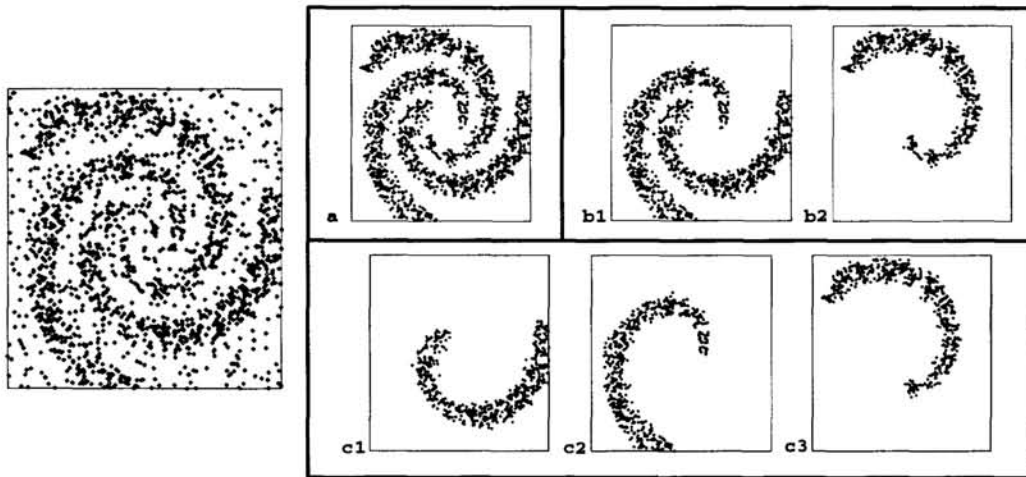

Figure 1: The 2000 data points (left), and the three most pronounced hierarchical levels of clustering (right). At $r=353$ the three spirals form one cluster (figure a). This cluster splits at $r=354$ into two (figures b1,b2), and into three parts at $r=368$ (figures c1,c2,c3). The background points form isolated clusters, usualy of size 1 (not shown).

Figure 1 shows 2000 data points in the Euclidean plane. In the stochastic stage of the algorithm we used only 200 iterations of graph contraction, during which we estimated for every pair $i$-$j$ the value of $r$ which satisfies $p_{ij}^r = 0.5$ (see Section 2.2).

As expected, subsequent partitions are typically identical or differ only slightly from each other (Figure 2). The variation between subsequent partitions was measured using the 10 largest parts ($K = 10$, see Section 2.3). The results did not depend on the exact value of $K$ since the sum was dominated by its first terms.

At low $r$ values (partition into a small number of components) a typical partition is composed of one giant component and a few tiny components that capture isolated noise points. The incorporation of these tiny components into the giant one produce negligible variations between subsequent partitions. At high $r$ values all the components are small, and therefore the variation between subsequent partitions must decay. At intermediate $r$ values a small number of sharp peaks appear.

The two highest peaks in Figure 2 are at $r=354$ and $r=368$; they mark meaningful hierarchies for the data clustering, as shown in Figure 1. We compare our results with two other methods in Figures 3 and 4.

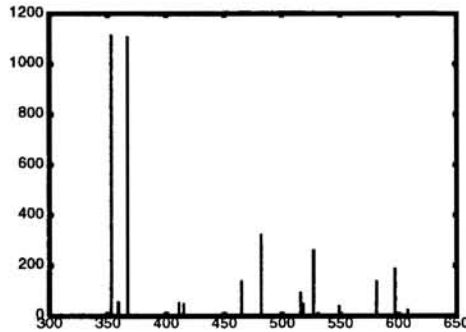

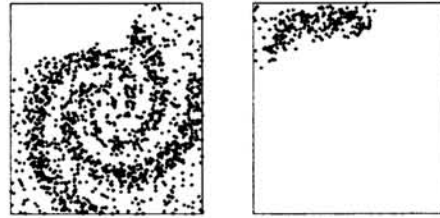

Figure 2: The variation between subsequent partitions (see text) as a function of the number of components ($r$). The variation is computed for every integer $r$ (the spacing between peaks is *not* due to sparse sampling). Outside the displayed range the variation vanishes.

Figure 3: The best bi-partition according to the normalized cut algorithm [9]. Since the first partition breaks one of the spirals, a satisfactory solution cannot be achieved in any of the later stages.

Figure 4: A three (macroscopic) clusters partition by a deterministic single linkage algorithm. The probabilistic scheme avoids the "bridging effect" thanks to the small probability of selecting the particular chain of edges.

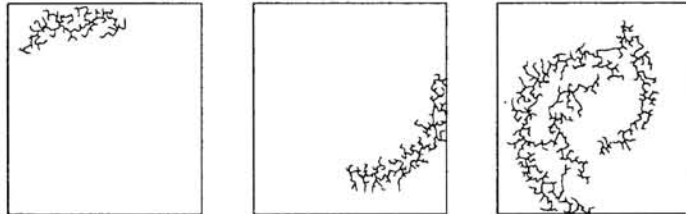

## Footnotes

[1]For example, the mutual neighborhood clustering algorithm [10] substitutes the edge weight $w_{ij}$ with a new weight $w'_{ij} = m + n$ where $i$ is the $m^{th}$ nearest neighbor of $j$ and $j$ is the $n^{th}$ nearest neighbor of $i$.

[2]The reader who is familiar with flow theory may notice that this algorithm also belongs to the first category of methods, as it is equivalent to a weight transformation followed by thresholding. The weight transformation replaces $w_{ij}$ by the maximal flow between $i$ and $j$.

[3]The exact decay rate is not known, but found experimentally to be adequate. Otherwise we would ignore cuts generated with high capacity.

[4]Thanks to Ido Bergman for pointing this out.

[5]The vectorial representation of data points is not useful even if it was available, since the parametric model is not known (see Section 1)

# References

[1] Blatt M., Wiseman S. and Domany E., "Data clustering using a model granular magnet", Neural Computation 9, 1805-1842, 1997.

[2] Duda O. and Hart E., *"Pattern classification and scene analysis"*, Wiley-Interscience, New York, 1973.

[3] Hofmann T. and Buhmann J., "Pairwise data clustering by deterministic annealing", PAMI 19, 1-14, 1997.

[4] Jain A. and Dubes R., *"Algorithms for clustering data"*, Prentice Hall, NJ, 1988.

[5] Karger D., "A new approach to the minimum cut problem", Journal of the ACM, 43(4) 1996.

[6] Klock H. and Buhmann J., "Data visualization by multidimensional scaling: a deterministic annealing approach", Technical Report IAI-TR-96-8, Institut fur Informatik III, University of Bonn. October 1996.

[7] Linial N., London E. and Rabinovich Y., "The geometry of graphs and some of its algorithmic applications", Combinatorica 15, 215-245, 1995.

[8] Rose K., Gurewitz E. and Fox G., "Constrained clustering as an optimization method", PAMI 15, 785-794, 1993.

[9] Shi J. and Malik J., "Normalized cuts and image segmentation", Proc. CVPR, 731-737, 1997.

[10] Smith S., "Threshold validity for mutual neighborhood clustering", PAMI 15, 89-92, 1993.

[11] Wu Z. and Leahy R., "An optimal graph theoretic approach to data clustering: theory and its application to image segmentation", PAMI 15, 1101-1113, 1993.
